# Advice Refinement in Knowledge-Based SVMs

**Gautam Kunapuli**
Univ. of Wisconsin-Madison
1300 University Avenue
Madison, WI 53705
kunapuli@wisc.edu

**Richard Maclin**
Univ. of Minnesota, Duluth
1114 Kirby Drive
Duluth, MN 55812
rmaclin@d.umn.edu

**Jude W. Shavlik**
Univ. of Wisconsin-Madison
1300 University Avenue
Madison, WI 53705
shavlik@cs.wisc.edu

## Abstract

Knowledge-based support vector machines (KBSVMs) incorporate advice from domain experts, which can improve generalization significantly. A major limitation that has not been fully addressed occurs when the expert advice is imperfect, which can lead to poorer models. We propose a model that extends KBSVMs and is able to not only learn from data and advice, but also simultaneously improves the advice. The proposed approach is particularly effective for knowledge discovery in domains with few labeled examples. The proposed model contains bilinear constraints, and is solved using two iterative approaches: successive linear programming and a constrained concave-convex approach. Experimental results demonstrate that these algorithms yield useful refinements to expert advice, as well as improve the performance of the learning algorithm overall.

## 1 Introduction

We are primarily interested in learning in domains where there is only a *small amount of labeled data* but *advice can be provided by a domain expert*. The goal is to refine this advice, which is usually only approximately correct, during learning, in such scenarios, to produce interpretable models that *generalize better* and *aid knowledge discovery*. For learning in complex environments, a number of researchers have shown that incorporating prior knowledge from experts can greatly improve the generalization of the model learned, often with many fewer labeled examples. Such approaches have been shown in rule-learning methods [16], artificial neural networks (ANNs) [21] and support vector machines (SVMs) [10, 17]. One limitation of these methods concerns how well they adapt when the knowledge provided by the expert is inexact or partially correct. Many of the rule-learning methods focus on rule refinement to learn better rules, while ANNs form the rules as portions of the network which are refined by backpropagation. Further, ANN methods have been paired with rule-extraction methods [3, 20] to try to understand the resulting learned network and provide rules that are easily interpreted by domain experts.

We consider the framework of knowledge-based support vector machines (KBSVMs), introduced by Fung et al. [6]. KBSVMs have been extensively studied, and in addition to linear classification, they have been extended to incorporate kernels [5], nonlinear advice [14] and for kernel approximation [13]. Recently, Kunapuli et al. derived an online version of KBSVMs [9], while other approaches such as that of Le et al. [11] modify the hypothesis space rather than the optimization problem. Extensive empirical results from this prior work establish that expert advice can be effective, especially for biomedical applications such as breast-cancer diagnosis. KBSVMs are an attractive methodology for knowledge discovery as they can produce good models that generalize well *with a small amount of labeled data*.

Advice tends to be rule-of-thumb and is based on the expert's accumulated experience in the domain; it may not always be accurate. Rather than simply ignoring or heavily penalizing inaccurate rules, the effectiveness of the advice can be *improved* through refinement. There are two main reasons for this: first, refined rules result in the improvement of the overall generalization, and second, if the refinements to the advice are interpretable by the domain experts, it will help in the understanding of the phenomena underlying the applications for the experts, and consequently

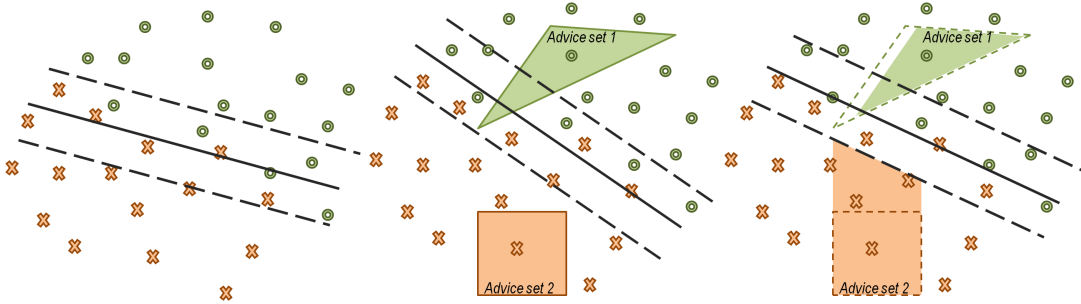

Figure 1: (**left**) Standard SVM, trades off complexity and loss wrt the data; (**center**) Knowledge-based SVM, also trades off loss wrt advice. A piece of advice set 1 extends over the margin, and is penalized as the advice error. No part of advice set 2 touches the margin, i.e., none of the rules in advice set 2 are useful as *support constraints*. (**right**) SVM that refines advice in two ways: (1) advice set 1 is refined so that no part of is on the wrong side of the optimal hyperplane, minimizing advice error, (2) advice set 2 is expanded until it touches the optimal margin thus maximizing coverage of input space.

greatly facilitate the knowledge-discovery process. This is the motivation behind this work. KB-SVMs already have several desirable properties that make them an ideal target for refinement. First, advice is specified as polyhedral regions in input space, whose constraints on the features are easily interpretable by non-experts. Second, it is well-known that KBSVMs can learn to generalize well with small data sets [9], and can even learn from advice alone [6]. Finally, owing to the simplicity of the formulation, advice-refinement terms for the rules can be incorporated directly into the model.

We further motivate advice refinement in KBSVMs with the following example. Figure 1 (left) shows an SVM, which trades off regularization with the data error. Figure 1 (center) illustrates KBSVMs in their standard form as shown in [6]. As mentioned before, expert rules are specified in the KBSVM framework as polyhedral *advice regions* in input space. They introduce a bias to focus the learner on a model that also includes the advice of the form $\forall x, (x \in \text{advice region } i) \Rightarrow class(x) = 1$. Advice regarding the regions for which $class(x) = -1$ can be specified similarly.

In the KBSVM (Figure 1, center), each advice region contributes to the final hypothesis in a KBSVM via its *advice vector*, $\mathbf{u}^1$ and $\mathbf{u}^2$ (as introduced in [6]; also see Section 2). The individual constraints that touch or intersect the margin have non-zero $u_j^i$ components. As a piece of advice region 1 extends beyond the margin, $\mathbf{u}^1 \neq 0$; furthermore, analogous to data error, this overlap is penalized as the *advice error*. As no part of advice set 2 touches the margin, $\mathbf{u}^2 = 0$ and none of its rules contribute anything to the final classifier. Again, analogous to support vectors, rules with non-zero $u_j^i$ components are called *support constraints* [6]. Consequently, in the final classifier the advice sets are incorporated with advice error (advice set 1) or are completely ignored (advice set 2). Even though the rules are inaccurate, they are able to improve generalization compared to the SVM. However, simply penalizing advice that introduces errors can make learning difficult as the user must carefully trade off between optimizing data or advice loss.

Now, consider an SVM that is capable of refining inaccurate advice (Figure 1, right). When advice is inaccurate and intersects the hyperplane, it is truncated such that it minimizes the advice error. Advice that was originally ignored is extended to cover as much of the input space as is feasible. The optimal classifier has now minimized the error with respect to the data and the refined advice and is able to further improve upon the performance of not just the SVM but also the KBSVM. Thus, the goal is to refine potentially inaccurate expert advice during learning so as to learn a model with the best generalization.

Our approach generalizes the work of Maclin et al. [12], to produce a model that corrects the polyhedral advice regions of KBSVMs. The resulting mathematical program is no longer a linear or quadratic program owing to *bilinear* correction factors in the constraints. We propose two algorithmic techniques to solve the resulting bilinear program, one based on successive linear programming [12], and the other based on a concave-convex procedure [24]. Before we describe advice refinement, we briefly introduce our notation and KBSVMs.

We wish to learn a linear classifier ($\mathbf{w}'\mathbf{x} = b$) given $\ell$ labeled data $(\mathbf{x}^j, y_j)_{j=1}^{\ell}$ with $\mathbf{x}^j \in \mathbb{R}^n$ and labels $y_j \in \{\pm 1\}$. Data are collected row-wise in the matrix $X \in \mathbb{R}^{\ell \times n}$, while $Y = \text{diag}(\mathbf{y})$ is the diagonal matrix of the labels. We assume that $m$ advice sets $(D_i, \mathbf{d}^i, z_i)_{i=1}^m$ are given in addition to the data (see Section 2), and if the $i$-th advice set has $k_i$ constraints, we have $D_i \in \mathbb{R}^{k_i \times n}$, $\mathbf{d}^i \in \mathbb{R}^{k_i}$ and $z_i = \{\pm 1\}$. The absolute value of a scalar $y$ is denoted $|y|$, the 1-norm of a vector $\mathbf{x}$

is denoted $\|\mathbf{x}\|_1 = \sum_{i=1}^{n} |x_i|$, and the *entrywise* 1-norm of a $m \times n$ matrix $A \in \mathbb{R}^{p \times q}$ is denoted $\|A\|_1 = \sum_{i=1}^{p} \sum_{i=1}^{q} |A_{ij}|$. Finally, $\mathbf{e}$ is a vector of ones of appropriate dimension.

## 2 Knowledge-Based Support Vector Machines

In KBSVMs, advice can be specified about *every* potential data point in the input space that satisfies certain advice constraints. For example, consider a task of learning to diagnose diabetes, based on features such as age, blood pressure, body mass index (`bmi`), plasma glucose concentration (`gluc`), etc. The National Institute for Health (NIH) provides the following guidelines to establish risk for Type-2 Diabetes[1]: a person who is obese (`bmi` $\geq 30$) with `gluc` $\geq 126$ is at strong risk for diabetes, while a person who is at normal weight (`bmi` $\leq 25$) with `gluc` $\leq 100$ is unlikely to have diabetes. This leads to two advice sets, one for each class:

$$(\texttt{bmi} \leq 25) \wedge (\texttt{gluc} \leq 100) \Rightarrow \neg\texttt{diabetes}; \quad (\texttt{bmi} \geq 30) \wedge (\texttt{gluc} \geq 126) \Rightarrow \texttt{diabetes}, \tag{1}$$

where $\neg$ is the negation operator. In general, rules such as the ones above define a polyhedral region of the input space and are expressed as the implication

$$D_i \mathbf{x} \leq \mathbf{d}^i \Rightarrow z_i(\mathbf{w}'\mathbf{x} - b) \geq 1, \tag{2}$$

where the *advice label* $z_i = +1$ indicates that all points $\mathbf{x}$ that satisfy the constraints for the $i$-th advice set, $D_i \mathbf{x} \leq \mathbf{d}^i$ belong to class $+1$, while $z = -1$ indicates the same for the other class. The standard linear SVM formulation (without incorporating advice) for binary classification optimizes *model complexity* $+ \lambda$ *data loss*:

$$\min_{\xi \geq 0, \mathbf{w}, b} \quad \|\mathbf{w}\|_1 + \lambda \mathbf{e}'\xi, \qquad \text{s.t.} \quad Y(X\mathbf{w} - \mathbf{e}b) + \xi \geq \mathbf{e}. \tag{3}$$

The implications (2), for the $i = 1, \ldots, m$, can be incorporated into (3) using the nonhomogeneous Farkas theorem of the alternative [6] that introduces advice vectors $\mathbf{u}^i$. The advice vectors perform the same role as the dual multipliers $\alpha$ in the classical SVM. Recall that points with non-zero $\alpha$'s are the *support vectors* which additively contribute to $\mathbf{w}$. Similarly, the constraints of an advice set which have non-zero $\mathbf{u}^i$'s are called *support constraints*. The resulting formulation is the KBSVM, which optimizes *model complexity* $+ \lambda$ *data loss* $+ \mu$ *advice loss*:

$$
\begin{aligned}
\min_{\mathbf{w}, b, (\xi, \mathbf{u}^i, \boldsymbol{\eta}^i, \zeta_i) \geq 0} \quad & \|\mathbf{w}\|_1 + \lambda \mathbf{e}'\xi + \mu \sum_{i=1}^{m} (\mathbf{e}'\boldsymbol{\eta}^i + \zeta_i) \\
\text{s.t.} \quad & Y(X\mathbf{w} - b\mathbf{e}) + \xi \geq \mathbf{e}, \\
& -\boldsymbol{\eta}^i \leq D_i'\mathbf{u}^i + z_i\mathbf{w} \leq \boldsymbol{\eta}^i, \\
& -\mathbf{d}^{i'}\mathbf{u}^i - z_i b + \zeta_i \geq 1, \; i = 1, \ldots, m.
\end{aligned}
\tag{4}
$$

In the case of inaccurate advice, the advice errors $\boldsymbol{\eta}^i$ and $\zeta_i$ soften the advice constraints analogous to the data errors $\xi$. Returning to Figure 1, for advice set 1, $\boldsymbol{\eta}^1$, $\zeta_1$ and $\mathbf{u}^1$ are non-zero, while for advice set 2, $\mathbf{u}^2 = 0$. The influence of data and advice is determined by the choice of the parameters $\lambda$ and $\mu$ which reflect the user's trust in the data and advice respectively.

## 3 Advice-Refining Knowledge-based Support Vector Machines

Previously, Maclin et al. [12] formulated a model to refine advice in KBSVMs. However, their model is limited as only the terms $\mathbf{d}^i$ are refined, which as we discuss below, greatly restricts the types of refinements that are possible. They only consider refinement terms $\mathbf{f}^i$ for the right hand side of the $i$-th advice set, and attempt to refine each rule such that

$$D_i \mathbf{x} \leq (\mathbf{d}^i - \mathbf{f}^i) \Rightarrow z_i(\mathbf{w}'\mathbf{x} - b) \geq 1, \; i = 1, \ldots, m. \tag{5}$$

The resulting formulation adds refinement terms into the KBSVM model (4) in the advice constraints, as well as in the objective. The latter allows for the overall extent of the refinement to be controlled by the *refinement parameter* $\nu > 0$. This formulation was called Refining-Rules Support Vector Machine (RRSVM):

$$
\begin{aligned}
\min_{\mathbf{w}, b, \mathbf{f}^i, (\xi, \mathbf{u}^i, \boldsymbol{\eta}^i, \zeta_i) \geq 0} \quad & \|\mathbf{w}\|_1 + \lambda \mathbf{e}'\xi + \mu \sum_{i=1}^{m} (\mathbf{e}'\boldsymbol{\eta}^i + \zeta_i) + \nu \sum_{i=1}^{m} \|\mathbf{f}^i\|_1 \\
\text{s.t.} \quad & Y(X\mathbf{w} - b\mathbf{e}) + \xi \geq \mathbf{e}, \\
& -\boldsymbol{\eta}^i \leq D_i'\mathbf{u}^i + z_i\mathbf{w} \leq \boldsymbol{\eta}^i, \\
& -(\mathbf{d}^i - \mathbf{f}^i)'\mathbf{u}^i - z_i b + \zeta_i \geq 1, \; i = 1, \ldots, m.
\end{aligned}
\tag{6}
$$

This problem is no longer an LP owing to the bilinear terms $\mathbf{f}^{i\prime}\mathbf{u}^i$ which make the refinement constraints non-convex. Maclin et al. solve this problem using successive linear programming (SLP) wherein linear programs arising from alternately fixing either the advice terms $\mathbf{d}^i$ or the refinement terms $\mathbf{f}^i$ are solved iteratively.

We consider a full generalization of the RRSVM approach and develop a model where it is possible to refine the entire advice region $D\mathbf{x} \leq \mathbf{d}$. This allows for much more flexibility in refining the advice based on the data, while still retaining interpretability of the resulting refined advice. In addition to the terms $\mathbf{f}^i$, we propose the introduction of additional refinement terms $F_i$ into the model, so that we can refine the rules in as general a manner as possible:

$$(D_i - F_i)\mathbf{x} \leq (\mathbf{d}^i - \mathbf{f}^i) \Rightarrow z_i(\mathbf{w}'\mathbf{x} - b) \geq 1, \ \ i = 1, \ldots, m. \tag{7}$$

Recall that for each advice set we have $D_i \in \mathbb{R}^{k_i \times n}$ and $\mathbf{d}^i \in \mathbb{R}^{k_i}$, i.e., the $i$-th advice set contains $k_i$ constraints. The corresponding refinement terms $F_i$ and $\mathbf{f}^i$ will have the same dimensions respectively as $D_i$ and $\mathbf{d}^i$. The formulation (6) now includes the additional refinement terms $F_i$, and the formulation optimizes:

$$
\begin{aligned}
\min_{\mathbf{w},b,F_i,\mathbf{f}^i,(\boldsymbol{\xi},\mathbf{u}^i,\boldsymbol{\eta}^i,\zeta_i)\geq 0} \quad & \|\mathbf{w}\|_1 + \lambda\mathbf{e}'\boldsymbol{\xi} + \mu \sum_{i=1}^m (\mathbf{e}'\boldsymbol{\eta}^i + \zeta_i) + \nu \sum_{i=1}^m \left( \|F_i\|_1 + \|\mathbf{f}^i\|_1 \right) \\
\text{s.t.} \quad & Y(X\mathbf{w} - b\mathbf{e}) + \boldsymbol{\xi} \geq \mathbf{e}, \\
& -\boldsymbol{\eta}^i \leq (D_i - F_i)'\mathbf{u}^i + z_i\mathbf{w} \leq \boldsymbol{\eta}^i, \\
& -(\mathbf{d}^i - \mathbf{f}^i)'\mathbf{u}^i - z_i b + \zeta_i \geq 1, \ \ i = 1, \ldots, m.
\end{aligned}
\tag{8}
$$

The objective function of (8) trades-off the effect of refinement in each of the advice sets via the *refinement parameter* $\nu$. This is the Advice-Refining KBSVM (arkSVM); it improves upon the work of Maclin et al. in two important ways. First, refining $\mathbf{d}$ alone is highly restrictive as it allows only for the *translation* of the boundaries of the polyhedral advice; the generalized refinement offered by arkSVMs allows for much more flexibility owing to the fact that the boundaries of the advice can be translated *and rotated* (see Figure 2). Second, the newly added refinement terms, $F_i'\mathbf{u}^i$, are bilinear also, and do not make the overall problem more complex; in addition to the successive linear programming approach of [12], we also propose a concave-convex procedure that leads to an approach based on successive quadratic programming. We provide details of both approaches next.

### 3.1 arkSVMs via Successive Linear Programming

One approach to solving bilinear programming problems is to solve a sequence of linear programs while alternately fixing the bilinear variables. This approach is called successive linear programming, and has been used to solve various machine learning formulations, for instance [1, 2]. In this approach, which was also adopted by [12], we solve the LPs arising from alternatingly fixing the sources of bilinearity: $(F_i, \mathbf{f}^i)_{i=1}^m$ and $\{\mathbf{u}^i\}_{i=1}^m$. Algorithm 1 describes the above approach. At the $t$-th iteration, the algorithm alternates between the following steps:

- (**Estimation Step**) When the refinement terms, $(\hat{F}_i^t, \hat{\mathbf{f}}^{i,t})_{i=1}^m$, are fixed the resulting LP becomes a standard KBSVM which attempts to find a data-estimate of the advice vectors $\{\mathbf{u}^i\}_{i=1}^m$ using the current refinement of the advice region: $(D_j - \hat{F}_j^t)\mathbf{x} \leq (\mathbf{d}^j - \hat{\mathbf{f}}^{j,t})$.

- (**Refinement Step**) When the advice-estimate terms $\{\hat{\mathbf{u}}^{i,t}\}_{i=1}^m$ are fixed, the resulting LP solves for $(F_i, \mathbf{f}^i)_{i=1}^m$ and attempts to further refine the advice regions based on estimates from data computed in the previous step.

**Proposition 1** *I. For $\epsilon = 0$, the sequence of objective values converges to the value* $\|\bar{\mathbf{w}}\|_1 + \lambda\mathbf{e}'\bar{\boldsymbol{\xi}} + \mu \sum_{i=1}^m (\mathbf{e}'\bar{\boldsymbol{\eta}}^i + \bar{\zeta}_i) + \nu \sum_{i=1}^m \left( \|\bar{F}_i\|_1 + \|\bar{\mathbf{f}}^i\|_1 \right)$, *where the data and advice errors* $(\bar{\boldsymbol{\xi}}, \bar{\boldsymbol{\eta}}^i, \bar{\zeta}_i)$ *are computed from any accumulation point* $(\bar{\mathbf{w}}, \bar{b}, \bar{\mathbf{u}}^i, \bar{F}_i, \bar{\mathbf{f}}^i)$ *of the sequence of iterates* $(\hat{\mathbf{w}}^t, \hat{b}^t, \hat{\mathbf{u}}^{i,t}, \hat{F}_i^t, \hat{\mathbf{f}}^{i,t})_{t=1}^\infty$ *generated by Algorithm 1.*

*II. Such an accumulation point satisfies the local minimum condition*

$$
\begin{aligned}
(\bar{\mathbf{w}}, \bar{b}) \in \min_{\substack{\mathbf{u}^i \geq 0 \\ \mathbf{w},b,(\boldsymbol{\xi},\boldsymbol{\eta}^i\zeta_i \geq 0)}} \quad & \|\mathbf{w}\|_1 + \lambda\mathbf{e}'\boldsymbol{\xi} + \mu \sum_{i=1}^m (\mathbf{e}'\boldsymbol{\eta}^i + \zeta_i) \\
\text{subject to} \quad & Y(X\mathbf{w} - b\mathbf{e}) + \boldsymbol{\xi} \geq \mathbf{e}, \\
& -\boldsymbol{\eta}^i \leq (D_i - \bar{F}_i)'\mathbf{u}^i + z_i\mathbf{w} \leq \boldsymbol{\eta}^i, \\
& -(\mathbf{d}^i - \bar{\mathbf{f}}^i)'\mathbf{u}^i - z_i b + \zeta_i \geq 1, \qquad i = 1, \ldots, m.
\end{aligned}
$$

---

**Algorithm 1** arkSVM via Successive Linear Programming (arkSVM-sla)

---

1: **initialize:** $t = 1, \hat{F}_i^1 = 0, \hat{\mathbf{f}}^{i,1} = 0$
2: **while feasible do**
3:     **if** $\mathbf{x}$ not feasible for $(D_i - \hat{F}_i^t)\,\mathbf{x} \leq (\mathbf{d}^j - \hat{\mathbf{f}}^{i,t})$      **return failure**
4:     (**estimation step**) solve for $\{\hat{\mathbf{u}}^{i,t+1}\}_{i=1}^m$

$$\min_{\mathbf{w},b,(\boldsymbol{\xi},\mathbf{u}^i,\boldsymbol{\eta}^i,\zeta_i)\geq 0} \quad \|\mathbf{w}\|_1 + \lambda \mathbf{e}'\boldsymbol{\xi} + \mu \sum_{i=1}^m (\mathbf{e}'\boldsymbol{\eta}^i + \zeta_i)$$
$$\text{s.t.} \quad Y(X\mathbf{w} - b\mathbf{e}) + \boldsymbol{\xi} \geq \mathbf{e},$$
$$-\boldsymbol{\eta}^i \leq (D_i - \hat{F}_i^t)'\mathbf{u}^i + z_i\mathbf{w} \leq \boldsymbol{\eta}^i,$$
$$-(\mathbf{d}^i - \hat{\mathbf{f}}^{i,t})'\mathbf{u}^i - z_i b + \zeta_i \geq 1, \quad i = 1, \dots, m.$$

5:     (**refinement step**) solve for $(\hat{F}_i^{t+1}, \hat{\mathbf{f}}^{i,t+1})_{i=1}^m$

$$\min_{\mathbf{w},b,F_i,\mathbf{f}^i,(\boldsymbol{\xi},\boldsymbol{\eta}^i,\zeta_i)\geq 0} \quad \|\mathbf{w}\|_1 + \lambda \mathbf{e}'\boldsymbol{\xi} + \mu \sum_{i=1}^m (\mathbf{e}'\boldsymbol{\eta}^i + \zeta_i) + \nu \sum_{i=1}^m \left( \|F_i\|_1 + \|\mathbf{f}^i\|_1 \right)$$
$$\text{s.t.} \quad Y(X\mathbf{w} - b\mathbf{e}) + \boldsymbol{\xi} \geq \mathbf{e},$$
$$-\boldsymbol{\eta}^i \leq (D_i - F_i)'\hat{\mathbf{u}}^{i,t+1} + z_i\mathbf{w} \leq \boldsymbol{\eta}^i,$$
$$-(\mathbf{d}^i - \mathbf{f}^i)'\hat{\mathbf{u}}^{i,t+1} - z_i b + \zeta_i \geq 1, \quad i = 1, \dots, m.$$

6:     (**termination test**) **if** $\sum_j \left( \|F_j^t - F_j^{t+1}\| + \|\mathbf{f}_j^t - \mathbf{f}_j^{t+1}\| \right) \leq \epsilon$      **then return** solution
7:     (**continue**) $t = t + 1$
8: **end while**

---

**Algorithm 2** arkSVM via Successive Quadratic Programming (arkSVM-sqp)

---

1: **initialize:** $t = 1, \hat{F}_i^1 = 0, \hat{\mathbf{f}}^{i,1} = 0$
2: **while feasible do**
3:     **if** $\mathbf{x}$ not feasible for $(D_i - \hat{F}_i^t)\,\mathbf{x} \leq (\mathbf{d}^j - \hat{\mathbf{f}}^{i,t})$      **return failure**
4:     solve for $\{\hat{\mathbf{u}}^{i,t+1}\}_{i=1}^m$

$$\min_{\substack{F_i,\mathbf{f}^i,(\mathbf{u}^i \geq 0) \\ \mathbf{w},b,(\boldsymbol{\xi},\boldsymbol{\eta}^i\zeta_i \geq 0)}} \quad \|\mathbf{w}\|_1 + \lambda \mathbf{e}'\boldsymbol{\xi} + \mu \sum_{i=1}^m (\mathbf{e}'\boldsymbol{\eta}^i + \zeta_i) + \nu \sum_{i=1}^m \left( \|F_i\|_1 + \|\mathbf{f}^i\|_1 \right)$$
$$\text{s.t.} \quad Y(X\mathbf{w} - b\mathbf{e}) + \boldsymbol{\xi} \geq \mathbf{e},$$
$$\text{eqns (10–12)}, \quad i = 1, \dots, m, \; j = 1, \dots, n$$

5:     (**termination test**) **if** $\sum_j \left( \|F_j^t - F_j^{t+1}\| + \|\mathbf{f}_j^t - \mathbf{f}_j^{t+1}\| \right) \leq \epsilon$      **then return** solution
6:     (**continue**) $t = t + 1$
7: **end while**

---

### 3.2   arkSVMs via Successive Quadratic Programming

In addition to the above approach, we introduce another algorithm (Algorithm 2) that is based on successive quadratic programming. In the constraint $(D_i - F_i)'\mathbf{u}^i + z_i\mathbf{w} - \boldsymbol{\eta}^i \leq 0$, only the refinement term $F_i'\mathbf{u}^i$ is bilinear, while the rest of the constraint is linear. Denote the $j$-th components of $\mathbf{w}$ and $\boldsymbol{\eta}^i$ to be $w_j$ and $\eta_j^i$ respectively. A general bilinear term $r's$, which is non-convex, can be written as the difference of two convex terms: $\frac{1}{4}\|r + s\|^2 - \frac{1}{4}\|r - s\|^2$. Thus, we have the equivalent constraint

$$D_{ij}'\mathbf{u}^i + z_i w_j - \eta_j^i + \frac{1}{4}\|F_{ij} - \mathbf{u}^i\|^2 \leq \frac{1}{4}\|F_{ij} + \mathbf{u}^i\|^2, \tag{9}$$

and both sides of the constraint above are convex and quadratic. We can linearize the right-hand side of (9) around some current estimate of the bilinear variables $(\hat{F}_{ij}^t, \hat{\mathbf{u}}^{i,t})$:

$$D_{ij}'\mathbf{u}^i + z_i w_j - \eta_j^i + \tfrac{1}{4}\|F_{ij} - \mathbf{u}^i\|^2 \leq \tfrac{1}{4}\|\hat{F}_{ij}^t + \hat{\mathbf{u}}^{i,t}\|^2$$
$$+ \tfrac{1}{2}(\hat{F}_{ij}^t + \hat{\mathbf{u}}^{i,t})' \left( (F_{ij} - \hat{F}_{ij}^t) + (\mathbf{u}^i - \hat{\mathbf{u}}^{i,t}) \right). \tag{10}$$

Similarly, the constraint $-(D_i - F_i)'\mathbf{u}^i - z_i\mathbf{w} - \boldsymbol{\eta}^i \leq 0$, can be replaced by

$$-D_{ij}'\mathbf{u}^i - z_i w_j - \eta_j^i + \tfrac{1}{4}\|F_{ij} + \mathbf{u}^i\|^2 \leq \tfrac{1}{4}\|\hat{F}_{ij}^t - \hat{\mathbf{u}}^{i,t}\|^2$$
$$+ \tfrac{1}{2}(\hat{F}_{ij}^t - \hat{\mathbf{u}}^{i,t})' \left( (F_{ij} - \hat{F}_{ij}^t) - (\mathbf{u}^i - \hat{\mathbf{u}}^{i,t}) \right), \tag{11}$$

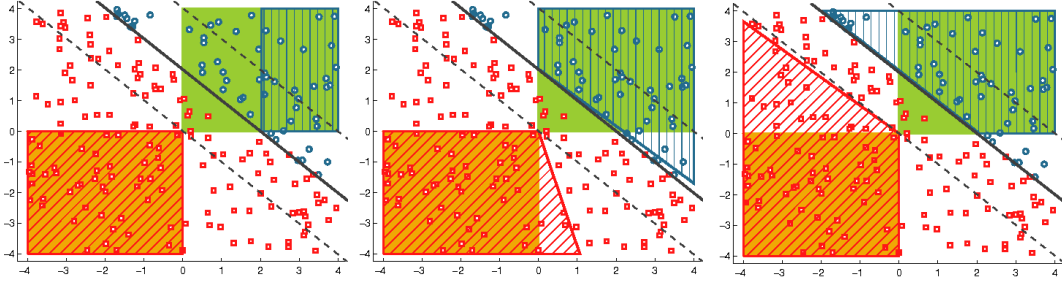

Figure 2: Toy data set (Section 4.1) using (**left**) RRSVM (**center**) arkSVM-sla (**right**) arkSVM-sqp. Orange and green unhatched regions show the original advice. The dashed lines show the margin, $\|\mathbf{w}\|_\infty$. For each method, we show the refined advice: vertically hatched for Class $+1$, and diagonally hatched for Class $-1$.

while $\mathbf{d}^{i'}\mathbf{u}^i + z_i b + 1 - \zeta_i - \mathbf{f}^{i'}\mathbf{u}^i \leq 0$ is replaced by

$$\mathbf{d}^{i'}\mathbf{u}^i + z_i b + 1 - \zeta_i + \tfrac{1}{4}\|\mathbf{f}^i - \mathbf{u}^i\|^2 \leq \tfrac{1}{4}\|\hat{\mathbf{f}}^{i,t} + \hat{\mathbf{u}}^{i,t}\|^2$$
$$+ \tfrac{1}{2}(\hat{\mathbf{f}}^{i,t} + \hat{\mathbf{u}}^{i,t})'\left((\mathbf{f}^{i,t} - \hat{\mathbf{f}}^{i,t}) + (\mathbf{u}^i - \hat{\mathbf{u}}^{i,t})\right). \quad (12)$$

The right-hand sides in (10–12) are affine and hence, the entire set of constraints are now convex. Replacing the original bilinear non-convex constraints of (8) with the convexified relaxations results in a quadratically-constrained linear program (QCLP). These quadratic constraints are more restrictive than their non-convex counterparts, which leads the feasible set of this problem to be a subset of that of the original problem. Now, we can iteratively solve the resulting QCLP. At the $t$-th iteration, the restricted problem uses the current estimate to construct a new feasible point and iterating this procedure produces a sequence of feasible points with decreasing objective values. The approach described here is essentially the constrained concave-convex procedure (CCCP) that has been discovered and rediscovered several times. Most recently, the approach was described in the context of machine learning approaches by Yuille and Rangarajan [24], and Smola and Vishwanathan [19], who also derived conditions under which the algorithm converges to a local solution. The following convergence theorem is due to [19].

**Proposition 2** *For Algorithm 2, the sequence of objective values converges to the value $\|\bar{\mathbf{w}}\|_1 + \lambda \mathbf{e}'\bar{\boldsymbol{\xi}} + \mu \sum_{i=1}^{m} (\mathbf{e}'\bar{\boldsymbol{\eta}}^i + \bar{\zeta}_i) + \nu \sum_{i=1}^{m} \left(\|\bar{F}_i\|_1 + \|\bar{\mathbf{f}}^i\|_1\right)$, where $(\bar{\mathbf{w}}, \bar{b}, \bar{\mathbf{u}}^i, \bar{F}_i, \bar{\mathbf{f}}^i, \bar{\boldsymbol{\xi}}, \bar{\boldsymbol{\eta}}^i, \bar{\zeta}_i)$ is the local minimum solution of (8) provided that the constraints (10–12) in conjunction with the convex constraints $Y(X\mathbf{w} - \mathbf{e}b) + \boldsymbol{\xi} \geq \mathbf{e}$, $\boldsymbol{\xi} \geq 0$, $\mathbf{u}^i \geq 0$, $\zeta_i \geq 0$ satisfy suitable constraint qualifications at the point of convergence of the algorithm.*

Both Algorithms 1 and 2 produce local minima solutions to the arkSVM formulation (8). For either solution, the following proposition holds, which shows that either algorithm produces a refinement of the original polyhedral advice regions. The proof is a direct consequence of [13][Proposition 2.1].

**Proposition 3** *Let $(\bar{\mathbf{w}}, \bar{b}, \bar{\mathbf{u}}^i, \bar{F}_i, \bar{\mathbf{f}}^i, \bar{\boldsymbol{\xi}}, \bar{\boldsymbol{\eta}}^i, \bar{\zeta}_i)$ be the local minimum solution produced by Algorithm 1 or Algorithm 2. Then, the following refinement to the advice sets holds:*

$$(D_i - \bar{F}_i) \leq (\mathbf{d}^i - \bar{\mathbf{f}}^i) \Rightarrow z_i(\bar{\mathbf{w}}'\mathbf{x} - \bar{b}) \geq -\hat{\boldsymbol{\eta}}^{i'}\mathbf{x} - \bar{\zeta}_i,$$

*where $-\bar{\boldsymbol{\eta}}^i \leq \hat{\boldsymbol{\eta}}^i \leq \bar{\boldsymbol{\eta}}^i$ such that $D_i'\bar{\mathbf{u}}^i + \bar{\mathbf{w}} + \hat{\boldsymbol{\eta}}^i = 0$.*

## 4 Experiments

We present the results of several experiments that compare the performance of three algorithms: RRSVMs (which only refine the $\mathbf{d}$ term in $D\mathbf{x} \leq \mathbf{d}$), arkSVM-sla (successive linear programming) and arkSVM-sqp (successive quadratic programming) with that of standard SVMs and KBSVMs. The LPs were solved using QSOPT[2], while the QCLPs were solved using SDPT-3 [22].

### 4.1 Toy Example

We illustrate the behavior of advice refinement algorithms discussed previously geometrically using a simple 2-dimensional example (Figure 2). This toy data set consists of 200 points separated by $x_1 + x_2 = 2$. There are two advice sets: $\{S_1 : (x_1, x_2) \geq 0 \Rightarrow z = +1\}$, $\{S_2 : (x_1, x_2) \leq 0 \Rightarrow$

[2]http://www2.isye.gatech.edu/~wcook/qsopt/

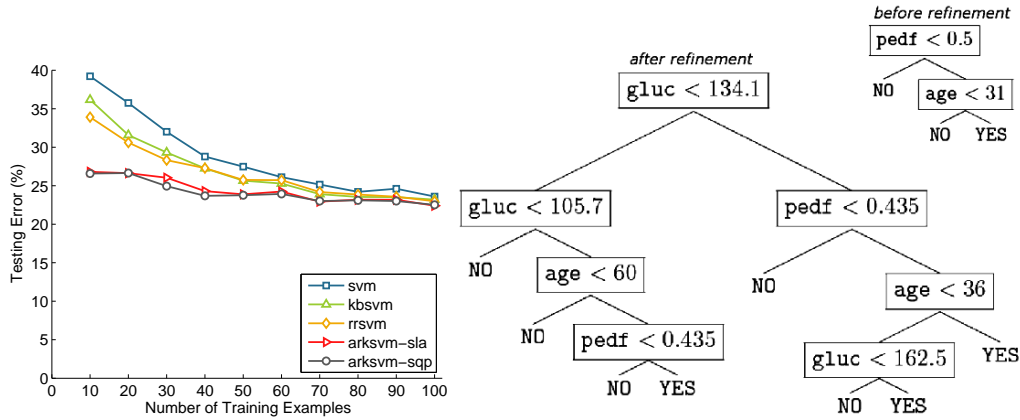

Figure 3: Diabetes data set, Section 4.2; (**left**) Results averaged over 10 runs on a hold-out test set of 412 points, with parameters selected by five-fold cross validation; (**right**) An approximate decision-tree representation of `Diabetes Rule 6` before and after refinement. The left branch is chosen if the query at a node is true, and the right branch otherwise. The leaf nodes classify the data point according to `?diabetes`.

$z = -1$}. Both arkSVMs are able to refine knowledge sets such that the no part of $S_1$ lies on the wrong side of the final hyperplane. In addition, the refinement terms allow for sufficient modification of the advice sets $D\mathbf{x} \le \mathbf{d}$ so that they fill the input space as much as possible, without violating the margin. Comparing to RRSVMs, we see that refinement is restrictive because corrections are applied only to part of the advice sets, rather than fully correcting the advice.

## 4.2 Case Study 1: PIMA Indians Diabetes Diagnosis

The Pima Indians Diabetes data set [4] has been studied for several decades and is used as a standard benchmark to test many machine learning algorithms. The goal is to predict the onset of diabetes in 768 Pima Indian women *within the next 5 years* based on current indicators (eight features): number of times pregnant, plasma glucose concentration (`gluc`), diastolic blood pressure, triceps skin fold test, 2-hour serum insulin, body mass index (`bmi`), diabetes pedigree function (`pedf`) and age. Studies [15] show that diabetes incidence among the Pima Indians is significantly higher among subjects with `bmi` $\ge 30$. In addition, a person with impaired glucose tolerance is at a significant risk for, or worse, has undiagnosed diabetes [8]. This leads to the following expert rules:

$$
\begin{array}{lll}
\text{(Diabetes Rule 1)} & (\texttt{gluc} \le 126) & \Rightarrow \neg\texttt{diabetes}, \\
\text{(Diabetes Rule 2)} & (\texttt{gluc} \ge 126) \wedge (\texttt{gluc} \le 140) \wedge (\texttt{bmi} \le 30) & \Rightarrow \neg\texttt{diabetes}, \\
\text{(Diabetes Rule 3)} & (\texttt{gluc} \ge 126) \wedge (\texttt{gluc} \le 140) \wedge (\texttt{bmi} \ge 30) & \Rightarrow \texttt{diabetes}, \\
\text{(Diabetes Rule 4)} & (\texttt{gluc} \ge 140) & \Rightarrow \texttt{diabetes}.
\end{array}
$$

The diabetes pedigree function was developed by Smith et al. [18], and uses genetic information from family relatives to provide a measure of the expected genetic influence (heredity) on the subject's diabetes risk. The function also takes into account the age of relatives who do have diabetes; on average, Pima Indians are only 36 years old[3] when diagnosed with diabetes. A subject with high heredity who is at least 31 is at a significantly increased risk for diabetes in the next five years:

$$
\begin{array}{lll}
\text{(Diabetes Rule 5)} & (\texttt{pedf} \le 0.5) \wedge (\texttt{age} \le 31) & \Rightarrow \neg\texttt{diabetes}, \\
\text{(Diabetes Rule 6)} & (\texttt{pedf} \ge 0.5) \wedge (\texttt{age} \ge 31) & \Rightarrow \texttt{diabetes}.
\end{array}
$$

Figure 3 (left) shows that unrefined advice does help initially, especially with as few as 30 data points. However, as more data points are available, the effect of the advice diminishes. In contrast, the advice refining methods are able to generalize much better with few data points, and eventually converge to a better solution. Finally, Figure 3 (right) shows an approximate tree representation of `Diabetes Rule 6` after refinement. This tree was constructed by sampling the space around refined advice region uniformly, and then training a decision tree that covers as many of the sampled points as possible. This naive approach to *rule extraction from refined advice* is shown here only to illustrate that it is possible to produce very useful domain-expert-interpretable rules from refinement. More efficient and accurate rule extraction techniques inspired by SVM-based rule extraction (for example, [7]) are currently under investigation.

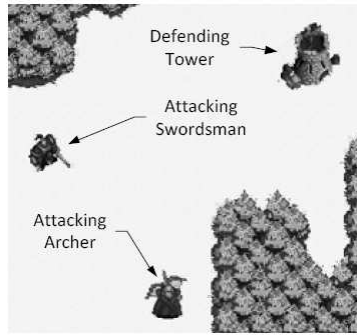 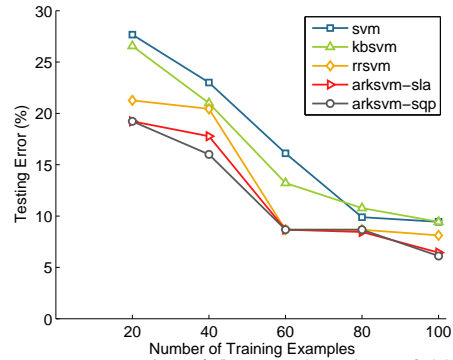

Figure 4: Wargus data set, Section 4.3; (**left**) An example Wargus scenario; (**right**) Results using 5-fold cross validation on a hold out test set of 1000 points.

## 4.3 Case Study 2: Refining GUI-Collected Human Advice in a Wargus Task

Wargus[4] is a real-time strategy game in which two or more players gather resources, build bases and control units in order to conquer opposing players. It has been widely used to study and evaluate various machine learning and planning algorithms. We evaluate our algorithms on a classification task in the Wargus domain developed by Walker et al. [23] called `tower-defense` (Figure 4, left). Advice for this task was collected from humans via a graphical, human-computer interface (HCI) as detailed in [23]. Each scenario (example) in `tower-defense`, consists of a single tower being attacked by a group of enemy units, and the task is to predict whether the tower will survive the attack and defeat the attackers given the size and composition of the latter, as well as other factors such as the environment. The data set consists of 80 features including information about units (eg., archers, ballista, peasants), unit properties (e.g., map location, health), group properties (e.g., `#archers`, `#footmen`) and environmental factors (e.g., `?hasMoat`).

Walker et al. [23] used this domain to study the feasibility of learning from human teachers. To this end, human players were first trained to identify whether a tower would fall given a particular scenario. Once the humans learned this task, they were asked to provide advice via a GUI-based interface based on specific examples. This setting lends itself very well to refinement as the advice collected from human experts represents the sum of their experiences with the domain, but is by no means perfect or exact. The following are some rules provided by human "domain experts":

$$(\text{Wargus Rule 1}) \quad (\#\texttt{footmen} \geq 3) \wedge (?\texttt{hasMoat} = 0) \qquad\qquad \Rightarrow \texttt{falls},$$
$$(\text{Wargus Rule 2}) \quad (\#\texttt{archers} \geq 5) \qquad\qquad\qquad\qquad\qquad\quad \Rightarrow \texttt{falls},$$
$$(\text{Wargus Rule 3}) \quad (\#\texttt{ballistas} \geq 1) \qquad\qquad\qquad\qquad\qquad\quad \Rightarrow \texttt{falls},$$
$$(\text{Wargus Rule 4}) \quad (\#\texttt{ballistas} = 0) \wedge (\#\texttt{archers} = 0) \wedge (?\texttt{hasMoat} = 1) \Rightarrow \texttt{stands}.$$

Figure 4 (right) shows the performance of the various algorithms on the Wargus data set. As with the previous case study, the arkSVM methods are able to not only learn very effectively with a small data set, they are also able to improve significantly on the performances of standard knowledge-based SVMs (KBSVMs) and rule-refining SVMs (RRSVMs).

## 5 Conclusions and Future Work

We have presented two novel knowledge-discovery methods: arkSVM-sla and arkSVM-sqp, that allow SVM methods to not only make use of advice provided by human experts but to *refine* that advice using labeled data to improve the advice. These methods are an advance over previous knowledge-based SVM methods which either did not refine advice [6] or could only refine simple aspects of the advice [12]. Experimental results demonstrate that our arkSVM methods can make use of inaccurate advice to revise them to better fit the data. A significant aspect of these learning methods is that the system not only produces a classifier but also produces human-inspectable changes to the user-provided advice, and can do so using small data sets. In terms of future work, we plan to explore several avenues of research including extending this approach to the nonlinear case for more complex models, better optimization algorithms for improved efficiency, and interpretation of refined rules for non-AI experts.

## Acknowledgements

The authors gratefully acknowledge support of the Defense Advanced Research Projects Agency under DARPA grant FA8650-06-C-7606 and the National Institute of Health under NLM grant R01-LM008796. Views and conclusions contained in this document are those of the authors and do not necessarily represent the official opinion or policies, either expressed or implied of the US government or of DARPA.

## Footnotes

[1] `http://diabetes.niddk.nih.gov/DM/pubs/~riskfortype2`

[3]`http://diabetes.niddk.nih.gov/dm/pubs/pima/kiddis/kiddis.htm`

[4]`http://wargus.sourceforge.net/index.shtml`

## References

[1] K. P. Bennett and E. J. Bredensteiner. A parametric optimization method for machine learning. *INFORMS Journal on Computing*, 9(3):311–318, 1997.

[2] K. P. Bennett and O. L. Mangasarian. Bilinear separation of two sets in n-space. *Computational Optimization and Applications*, 2:207–227, 1993.

[3] M. W. Craven and J. W. Shavlik. Extracting tree-structured representations of trained networks. In *Advances in Neural Information Processing Systems*, volume 8, pages 24–30, 1996.

[4] A. Frank and A. Asuncion. UCI machine learning repository, 2010.

[5] G. Fung, O. L. Mangasarian, and J. W. Shavlik. Knowledge-based nonlinear kernel classifiers. In *Sixteenth Annual Conference on Learning Theory*, pages 102–113, 2003.

[6] G. Fung, O. L. Mangasarian, and J. W. Shavlik. Knowledge-based support vector classifiers. In *Advances in Neural Information Processing Systems*, volume 15, pages 521–528, 2003.

[7] G. Fung, S. Sandilya, and R. B. Rao. Rule extraction from linear support vector machines. In *Proc. Eleventh ACM SIGKDD Intl. Conference on Knowledge Discovery in Data Mining*, pages 32–40, 2005.

[8] M. I. Harris, K. M. Flegal, C. C. Cowie, M. S. Eberhardt, D. E. Goldstein, R. R. Little, H. M. Wiedmeyer, and D. D. Byrd-Holt. Prevalence of diabetes, impaired fasting glucose, and impaired glucose tolerance in U.S. adults. *Diabetes Care*, 21(4):518–524, 1998.

[9] G. Kunapuli, K. P. Bennett, A. Shabbeer, R. Maclin, and J. W. Shavlik. Online knowledge-based support vector machines. In *Proc. of the European Conference on Machine Learning*, pages 145–161, 2010.

[10] F. Lauer and G. Bloch. Incorporating prior knowledge in support vector machines for classification: A review. *Neurocomputing*, 71(7–9):1578–1594, 2008.

[11] Q. V. Le, A. J. Smola, and T. Gärtner. Simpler knowledge-based support vector machines. In *Proceedings of the Twenty-Third International Conference on Machine Learning*, pages 521–528, 2006.

[12] R. Maclin, E. W. Wild, J. W. Shavlik, L. Torrey, and T. Walker. Refining rules incorporated into knowledge-based support vector learners via successive linear programming. In *AAAI Twenty-Second Conference on Artificial Intelligence*, pages 584–589, 2007.

[13] O. L. Mangasarian, J. W. Shavlik, and E. W. Wild. Knowledge-based kernel approximation. *Journal of Machine Learning Research*, 5:1127–1141, 2004.

[14] O. L. Mangasarian and E. W. Wild. Nonlinear knowledge-based classification. *IEEE Transactions on Neural Networks*, 19(10):1826–1832, 2008.

[15] M. E. Pavkov, R. L. Hanson, W. C. Knowler, P. H. Bennett, J. Krakoff, and R. G. Nelson. Changing patterns of Type 2 diabetes incidence among Pima Indians. *Diabetes Care*, 30(7):1758–1763, 2007.

[16] M. Pazzani and D. Kibler. The utility of knowledge in inductive learning. *Mach. Learn.*, 9:57–94, 1992.

[17] B. Schölkopf, P. Simard, A. Smola, and V. Vapnik. Prior knowledge in support vector kernels. In *Advances in Neural Information Processing Systems*, volume 10, pages 640–646, 1998.

[18] J. W. Smith, J. E. Everhart, W. C. Dickson, W. C. Knowler, and R. S. Johannes. Using the ADAP learning algorithm to forecast the onset of diabetes mellitus. In *Proc. of the Symposium on Comp. Apps. and Medical Care*, pages 261–265. IEEE Computer Society Press, 1988.

[19] A. J. Smola and S. V. N. Vishwanathan. Kernel methods for missing variables. In *Proceedings of the Tenth International Workshop on Artificial Intelligence and Statistics*, pages 325–332, 2005.

[20] S. Thrun. Extracting rules from artificial neural networks with distributed representations. In *Advances in Neural Information Processing Systems*, volume 8, 1995.

[21] G. G. Towell and J. W. Shavlik. Knowledge-based artificial neural networks. *Artificial Intelligence*, 70(1–2):119–165, 1994.

[22] R. H. Tütüncü, K. C. Toh, and M. J. Todd. Solving semidefinite-quadratic-linear programs using SDPT3. *Mathematical Programming*, 95(2), 2003.

[23] T. Walker, G. Kunapuli, N. Larsen, D. Page, and J. W. Shavlik. Integrating knowledge capture and supervised learning through a human-computer interface. In *Proc. Fifth Intl. Conf. Knowl. Capture*, 2011.

[24] A. L. Yuille and A. Rangarajan. The concave-convex procedure (CCCP). In *Advances in Neural Information Processing Systems*, volume 13, 2001.

